# Hidden Markov Model of Cortical Synaptic Plasticity: Derivation of the Learning Rule

**Michael Eisele**
W. M. Keck Center
for Integrative Neuroscience
San Francisco, CA 94143-0444
eisele@phy.ucsf.edu

**Kenneth D. Miller**
W. M. Keck Center
for Integrative Neuroscience
San Francisco, CA 94143-0444
ken@phy.ucsf.edu

## Abstract

Cortical synaptic plasticity depends on the relative timing of pre- and postsynaptic spikes and also on the temporal pattern of presynaptic spikes and of postsynaptic spikes. We study the hypothesis that cortical synaptic plasticity does not associate individual spikes, but rather whole firing episodes, and depends only on when these episodes start and how long they last, but as little as possible on the timing of individual spikes. Here we present the mathematical background for such a study. Standard methods from hidden Markov models are used to define what "firing episodes" are. Estimating the probability of being in such an episode requires not only the knowledge of past spikes, but also of future spikes. We show how to construct a causal learning rule, which depends only on past spikes, but associates pre- and postsynaptic firing episodes as if it also knew future spikes. We also show that this learning rule agrees with some features of synaptic plasticity in superficial layers of rat visual cortex (Froemke and Dan, Nature 416:433, 2002).

## 1 Introduction

Cortical synaptic plasticity agrees with the Hebbian learning principle: Neurons that fire together, wire together. But many features of cortical plasticity go beyond this simple principle, such as the dependence on spike-timing or the nonlinear dependence on spike frequency (see [1] or [2] for review). Studying these features may produce a better understanding of which neurons wire together in the neocortex.

Previous models of cortical synaptic plasticity [3]-[5] differed in their details, but they agreed that nonlinear learning rules are needed to model cortical plasticity. In linear learning rules, the weight change induced by a presynatic spike would depend only on the postsynaptic spikes, but not on all the other presynaptic spikes. In the cortex, by contrast, the contribution from a presynaptic spike is stronger when it occurs alone than when it occurs right after another presynaptic spike [5]. Similar results hold for postsynaptic spikes. Consequently, the weight change depends in a complex way on the whole temporal pattern of pre- and postsynaptic spikes. Even though this nonlinear dependence can be modeled phenomenologically [3]-[5], its biological function remains unknown. We will not propose such a function here, but reduce this complex dependence to a few principles, whose

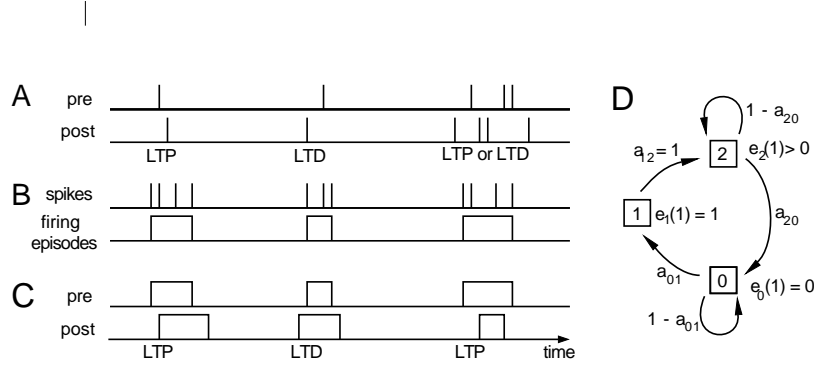

Figure 1: A: Usually, models of cortical synaptic plasticity associate pre- and postsynaptic spikes directly. They produce long-term potentiation (LTP) when the presynaptic spike (pre) precedes the postsynaptic spike (post), and long-term depression (LTD) if the order is reversed. When several pre- and postsynaptic spikes are interleaved in time, the outcome depends in a complicated way on the whole spike pattern (LTP or LTD). B: In our model, pre- and postsynaptic spikes are paired only indirectly. Each spike train is used to estimate when firing episodes start and end. C: These firing episodes are then associated, with LTP being induced if the presynaptic firing episode starts before the postsynaptic one and LTD if the order is reversed and if the episodes are short. D: Hidden Markov model used to estimate when firing episodes occur.

function may be easier to understand in future studies.

## 2   Basic learning principle

The basic principle behind our model is illustrated in fig. 1. We propose that the learning rule does not associate pre- and postsynaptic spikes directly, but rather uses them to estimate whether the pre- or postsynaptic neuron is currently in a period of rapid firing ('firing episode') or a period of little or no firing. It then associates the firing episodes. When the per- and postsynaptic firing episodes overlap, the synapse is strengthened or weakened depending on which one started first, but independent of the precise temporal patterns of spikes within a firing episode. As a consequence, the contribution of each spike to synaptic plasticity will depend on whether it occurs alone, or surrounded by other spikes, and the learning rule will be nonlinear. For the right parameter choice, the nonlinear features of this rule will agree well with nonlinear features of cortical synaptic plasticity.

Implementation of this rule will be done in two steps. Firstly, we will define what "firing episodes" are. Secondly, we will associate the pre- and postsynaptic firing episodes. The first step uses standard methods from hidden Markov models (see e.g. [6]). The pre- and postsynaptic neuron will each be described by a Markov model with three states (fig. 1D), which correspond to firing episodes (state 2; firing probability $e_2(1) > 0$), to the silence between responses (state 0; firing probability $e_0(1) = 0$), and to the first spike of a new firing episode (state 1; firing probability $e_1(1) = 1$; duration = 1 time step). As usual, the parameters of the Markov model are the transition probabilities $a_{kl}$, which determine how long firing episodes and silent periods are expected to last, and the emission rates $e_l(x_i)$, which determine the firing rates. $x_i$ is the binary observable at time step $i$ ($x_i = 1$ at spikes and $x_i = 0$ otherwise), $e_l(1)$ is the firing probability per time step in state $l$, and $e_l(0) = 1 - e_l(1)$. In general, the pre- and postsynaptic neuron will have different parameters $e_l(x)$ and $a_{kl}$.

Once the Markov model is defined, one can use standard algorithms (forward and backward algorithm) to estimate, for any given spike sequence, the state probabilities over time. To model cortical synaptic plasticity, we will increase the synaptic weight whenever the pre- and the postsynaptic neuron have simultaneous firing episodes (both in state 2), and decrease the weight whenever the postsynaptic firing episode starts first (pre in state 1 while post already in state 2):

$$\Delta w(\pi_i^{pre}, \pi_i^{post}) = \begin{cases} A^+ & \text{for } \pi_i^{pre} = 2,\ \pi_i^{post} = 2 \\ -A^- & \text{for } \pi_i^{pre} = 1,\ \pi_i^{post} = 2 \\ 0 & \text{otherwise} \end{cases} \tag{1}$$

where $A^+$ and $A^-$ are the amplitudes of synaptic potentiation and depression. In general, the states are not known with certainty, only their probabilities are, and the actual weight change is therefore defined as:

$$\sum_{h,l} \Delta w(h,l) \cdot P(\pi_i^{pre} = h \mid x^{pre}) \cdot P(\pi_i^{post} = l \mid x^{post}) \tag{2}$$

where the sum is over all possible pre- and postsynaptic states and $P(...|x)$ is the probability given the whole spike sequence $x_1, x_2, x_3, ....$ As fig. 2 shows, this straightforward learning rule produces weight changes that are similar to those seen in cortex [5]. (One can show that this particular Markov model depends on the parameters $a$ and $e$ only through the two combinations $\tau = dt/(e_2(1) + a_{20} - a_{01})$ and $\mu = dt \cdot e_2(1)/(a_{20} \cdot a_{01})$ where $dt$ is the time step. To fit the data on spike pairs and triplets [5], we set $\tau^{pre} = 15$ms, $\tau^{post} = 34$ms, $\mu^{pre} = 20$ms, $\mu^{post} = 70$ms, $A^+ = 96$Hz$\cdot dt$, and $A^- = 1.5$.)

This learning rule is, however, not biologically plausible, because it violates causality. The estimates of state probabilities depend not only on past, but also on future observables, while real synaptic plasticity can depend only on past spikes. To solve this causality problem, we will rewrite the learning rule, essentially deriving a new algorithm in place of the familiar hidden Markov algorithms. We will derive this causal learning rule not only for this specific 3-state model, but for general Markov models.

## 3 General form of the learning rule

### 3.1 Learning goal

To derive the general form of the learning rule for arbitrary pre- and postsynaptic Markov models, we assume that the transition probabilities $a_{kl}$ and emission probabilities $e_l(x_i)$ are given and that the weight change is some function

$$\Delta w\left(\pi_i^{pre}, \pi_i^{post}, i\right) \tag{3}$$

of the pre- and postsynaptic states $\pi_i$ at time $i$ and the time $i$ itself. If the pre- and postsynaptic state sequences $\pi^{pre}$ and $\pi^{post}$ were known, the weight $w_i$ at time $i$ would simply be the initial weight $w_0$ plus all the previous weight changes:

$$w_i\left[\pi^{pre}, \pi^{pre}\right] = w_0 + \sum_{j=1}^{i} \Delta w\left(\pi_j^{pre}, \pi_j^{post}, j\right) \tag{4}$$

In the current context, the state sequences are unknown and have to be estimated from the spike trains $x^{pre}$ and $x^{post}$. Ideally, we would like to set the weight at time $i$ equal to the expectation value of $w_i\left[\pi^{pre}, \pi^{post}\right]$, given the spike trains $x^{pre}$ and $x^{post}$. But only part of these spike trains are known at time $i$. Of the sequence $x^{pre}$ the synapse has already seen the past values $x_1^{pre}, x_2^{pre} ... x_{i-1}^{pre}$, which we will call $x_-^{pre}$, and the present value $x_i^{pre}$. But

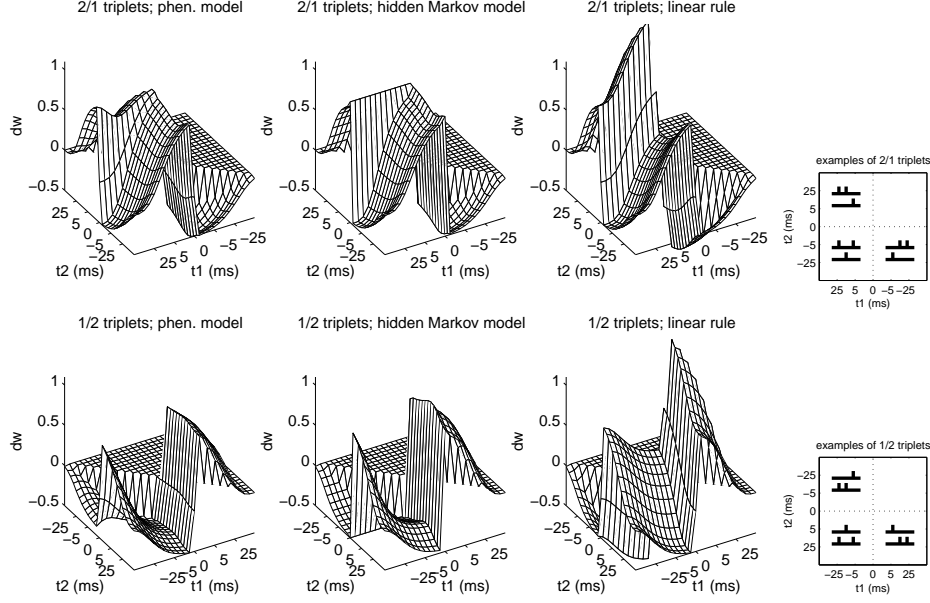

Figure 2: Weight change produced by spike triplets in various models. Our learning rule (second column), which depends on the timing of firing episodes but only weakly on the timing of individual spikes, and which was implemented using hidden Markov models, agrees well with the phenomenological model (first column) that was used in [5, fig 3b] to fit data from superficial layers in rat visual cortex. It certainly agrees better than a purely linear rule (third column). Parameters were set so that all three models produce the same results for spike pairs (1 presynaptic and 1 postsynaptic spike). Upper row: Weight change produced by 2 presynaptic and 1 postsynaptic spikes (2/1 triplet). Lower row: 1 presynaptic and 2 postsynaptic spikes (1/2 triplet). $t1$ and $t2$ are the times between pre- and postsynaptic spikes. The small boxes on the right show examples of spike patterns for positive and negative $t_1$ and $t_2$

it has not yet seen the future sequence $x_{i+1}^{pre}$, $x_{i+2}^{pre}$, ..., which we will call $x_+^{pre}$. All one can do is to make some assumption about what the future spikes will be, set $w_i$ accordingly, and correct $w_i$ in the future, when the real spike sequence becomes known. Our algorithm assumes no future spikes and sets the weight at time $i$ equal to:

$$ w_i = E\left\{ w_i\left[\pi^{pre}, \pi^{pre}\right] \mid x_+^{pre} = 0, x_i^{pre}, x_-^{pre}, x_+^{post} = 0, x_i^{post}, x_-^{post} \right\} \qquad (5) $$

where $E(...|x\}$ is the expectation value given the spike sequences $x$. The condition that all future spikes are 0 is written as $x_+^{pre} = 0$ and $x_+^{post} = 0$. One could make other assumptions about the future spikes, but all these assumptions would affect only when the weight changes, but not how much it changes in the long run. This is because the expectation value of a past weight change:

$$ E\left\{ \Delta w\left(\pi_j^{pre}, \pi_j^{post}, j\right) \mid x_+^{pre}, x_i^{pre}, x_-^{pre}, x_+^{post}, x_i^{post}, x_-^{post} \right\} \qquad (6) $$

will depend little on the future spikes $x_+^{pre}$ and $x_+^{post}$, if the time $j$ is much earlier than the time $i$. As $i$ grows, most weight changes will lie in the distant past and depend only weakly on our assumptions about future spikes.

Next we will show how to compute the expectation value in eq. (5) without having to store the past spike trains $x_-$. To simplify the notation, we will regard each pair of pre- and

postsynaptic states $\left(\pi_j^{pre}, \pi_j^{post}\right)$ as a state $\pi_j$ of a combined pre- and postsynaptic Markov model. We will also combine the pre- and postsynaptic spikes $\left(x_j^{pre}, x_j^{post}\right)$, each of which can take the two values 0 or 1, to a single observable $x_j$, which can take 4 values. The desired weight is then equal to:

$$w_i = E\left\{ w_i\left[\pi\right] \mid x_+ = 0,\, x_i,\, x_- \right\} \quad \text{with} \quad w_i\left[\pi\right] = w_0 + \sum_{j=1}^{i} \Delta w\left(\pi_j, j\right) \qquad (7)$$

### 3.2 Running estimate of state probabilities

To compute $w_i$, it is helpful to first compute the probabilities

$$q_l\left(i\right) = P\left(\pi_i = l \mid x_+ = 0,\, x_i,\, x_-\right) \qquad (8)$$

of the states given the past and present spikes and assuming that there are no future spikes. The $q_l\left(i\right)$ can be computed recursively, in terms of $q_k\left(i-1\right)$ (this is similar to the familiar forward algorithm for hidden Markov models). Write $q$ as:

$$q_l\left(i\right) = \sum_k P\left(\pi_i = l\,, \pi_{i-1} = k \mid x_+ = 0,\, x_i,\, x_-\right) \qquad (9)$$

$$= \sum_k P\left(x_+ = 0,\, x_i,\, \pi_i = l\,, \pi_{i-1} = k,\, x_-\right) / P\left(x_+ = 0,\, x_i,\, x_-\right) \qquad (10)$$

Because of the Markov property, future and present spikes $x_+$ and $x_i$ depend only on the present state $\pi_i$, but not on the past state $\pi_{i-1}$ or on $x_-$. Similarly, $\pi_i$ depends only on $\pi_{i-1}$ but not on $x_-$. Thus the enumerator of the last expression is equal to:

$$P\left(x_+ = 0 \mid \pi_i = l\right) \cdot P\left(x_i \mid \pi_i = l\right) \cdot P\left(\pi_i = l \mid \pi_{i-1} = k\right) \cdot P\left(\pi_{i-1} = k,\, x_-\right) \qquad (11)$$

$$= o_l(i) \cdot e_l(x_i) \cdot a_{kl} \cdot P\left(\pi_{i-1} = k,\, x_-\right) \qquad (12)$$

$$\text{with} \quad o_l(i) = P\left(x_+ = 0 \mid \pi_i = l\right) \qquad (13)$$

The probabilities $o_l(i)$ of having no future spikes after state $l$ can be computed by the backward algorithm:

$$o_l(i) = \sum_h P\left(x_+ = 0,\, \pi_{i+1} = h \mid \pi_i = l\right) = \sum_h o_h(i+1) \cdot e_h(0) \cdot a_{lh} \qquad (14)$$

This is a linear equation with constant coefficients. As long as the end of the Markov chain is far enough in the future, this equation reduces to an eigenvalue problem with the solution $o_l(i) = \lambda \cdot o_l(i+1)$, where $\lambda$ is the largest eigenvalue of the matrix with elements $e_h(0) \cdot a_{lh}$ and $o$ is the corresponding eigenvector. As the matrix elements are positive, $\lambda$ will be real, and the eigenvector will be unique up to a constant factor (except for quite exceptional, disconnected Markov chains, in which it may depend on the choice of end state). The last unknown factor in eq. (12) is $P\left(\pi_{i-1} = k\,,\, x_-\right)$, which can be expressed in terms of $q_k(i-1)$:

$$P\left(\pi_{i-1} = k,\, x_-\right) = q_k(i-1)\, P\left(x_+ = 0,\, x_i = 0,\, x_-\right) / P\left(x_+ = 0,\, x_i = 0 \mid \pi_{i-1} = k\right) \qquad (15)$$

where the Markov property was used again. Putting everything together, one gets the update rule for $q_l(i)$:

$$q_l(i) = \sum_k m_{kl}\left(x_i, x_-\right) \cdot q_k(i-1) \qquad (16)$$

$$\text{with} \quad m_{kl}\left(x_i, x_-\right) = n\left(x_i, x_-\right) \cdot e_l(x_i) \cdot a_{kl} \cdot o_l(i) / o_k(i-1) \qquad (17)$$

$$n\left(x_i, x_-\right) = P\left(x_+ = 0,\, x_i = 0,\, x_-\right) / P\left(x_+ = 0,\, x_i,\, x_-\right) \qquad (18)$$

The ratio $o_l(i)/o_k(i-1) = o_l(i)/(\lambda \cdot o_k(i))$ does not really depend on $i$ but only on the eigenvalue $\lambda$ and the relative size of the elements of the eigenvector $o$. If there is no pre- or postsynaptic spike at time $i$ $(x_i = 0)$, the normalization factor $n(x_i, x_-)$ is equal to 1, and $m_{kl}$ no longer depends on $i$ or $x_-$. In this case, eq. (16) is a linear equation with constant coefficients, which can be integrated analytically from one spike to the next, thereby speeding up the numerical simulation. At pre- or postsynaptic spikes $(x_i \neq 0)$, $n$ can be computed by summing eq. (16) over $l$ and using $\sum_l q_l(i) = 1$:

$$n(x_i, x_-) = \left( \sum_{l,k} q_k(i-1) \cdot e_l(x_i) \cdot a_{kl} \cdot o_l(i)/o_k(i-1) \right)^{-1} \tag{19}$$

### 3.3 Running estimate of weights

Using the knowledge of the probabilities $q_l(i)$, one can now compute the weight

$$w_i = E\{w_i[\pi] \mid x_+ = 0, x_i, x_-\} \tag{20}$$

$$= E\{w_{i-1}[\pi] \mid x_+ = 0, x_i, x_-\} + \sum_l \Delta w(l,i) \cdot q_l(i) \tag{21}$$

The expectation value $E\{w_{i-1}[\pi] \mid x\}$ in this equation will be equal to $w_{i-1}$, if there is no pre- or postsynaptic spike at time $i$ $(x_i = 0)$. In between spikes, the weight therefore changes as:

$$w_i = w_{i-1} + \sum_l \Delta w(l,i) \cdot q_l(i) \tag{22}$$

At the time of spikes, the weight change is more complex, because earlier weight changes have to be modified according to the new state information given by the spikes. To compute it, let us introduce the quantities

$$u_l(i) = q_l(i) \cdot E\{w_i[\pi] \mid x_+ = 0, x_i, \pi_i = l, x_-\} \tag{23}$$

The weight is equal to the sum of these $u$:

$$w_i = \sum_l u_l(i) \tag{24}$$

and, as we will see next, the $u_l(i)$ can be computed in a recursive way, even in the presence of spikes. Start with:

$$u_l(i) = q_l(i) \cdot (\Delta w(l,i) + E\{w_{i-1}[\pi] \mid x_+ = 0, x_i, \pi_i = l, x_-\}) \tag{25}$$

$$= \Delta w(l,i) \cdot q_l(i) + \sum_k P(\pi_{i-1} = k \mid x_+ = 0, x_i, \pi_i = l, x_-) \cdot q_l(i) \cdot$$

$$\cdot E\{w_{i-1}[\pi] \mid x_+ = 0, x_i, \pi_i = l, \pi_{i-1} = k, x_-\} \tag{26}$$

Because of the Markov property, the last expectation value depends only on $x_-$ and $k$, but not on $x_i$, $l$, or $x_+$, and it is thus equal to $u_k(i-1)/q_k(i-1)$. The other two factors

$$P(\pi_{i-1} = k \mid x_+ = 0, x_i, \pi_i = l, x_-) \cdot q_l(i) = P(\pi_i = l, \pi_{i-1} = k \mid x_+ = 0, x_i, x_-) \tag{27}$$

combine to give the same expression that already occurred in equation (9). As shown above (eq. (16)), this expression is equal to

$$m_{kl}(x_i, x_-) \cdot q_k(i-1) \tag{28}$$

with the same $m_{kl}$ as before. Putting everything together, one gets the update rule for $u_l(i)$:

$$u_l(i) = \Delta w(l,i) \cdot q_l(i) + \sum_k m_{kl}(x_i, x_-) \cdot u_k(i-1) \tag{29}$$

Together with eqs. (16), (17), (19), and (24) this constitutes our learning rule. It is causal, because it depends only on past, not on future signals, but in the long run it will give the same weight change as the standard hidden Markov rule (2). In between spikes, the $q$ in eq. (16) and the $u$ in eq. (29) evolve according to linear rules, and the weight changes according to the simple rule (22). These simplifications are a consequence of assuming, in the definition of $w_i$, that there are no future spikes. Other assumptions are possible: One could, for example, set $w_i$ equal to $w_i = E\{w_i[\pi] \mid x_i, x_-\}$, assuming that future spikes occur with the rate predicted by the Markov model, and one could also derive a causal learning rule for this $w_i$ (not shown), but then the evolution of $q$ and $u$ between spikes would be nonlinear and the evolution of $w$ would also be more complex.

This learning rule still has a rather unusual form. Usually, one writes $w_i$ as the sum of $w_{i-1}$ plus some weight change. Our rule can also be written in this form, if the $u$ are replaced by:

$$
\begin{aligned}
d_l(i) &= u_l(i) - q_l(i) \cdot w_i && (30) \\
&= q_l(i)\left(E\{w_i[\pi] \mid x_+ = 0, x_i, \pi_i = l, x_-\} - E\{w_i[\pi] \mid x_+ = 0, x_i, x_-\}\right) && (31)
\end{aligned}
$$

$d_l(i)$ is a measure for how much the weight should be changed if one suddenly learned, with certainty, that the neurons are in state $l$. By definition, the $d$ sum to zero: $\sum_l d_l(i) = 0$. Inserting the update rule for $u_l(i)$ gives the update rule for $d_l(i)$:

$$
\begin{aligned}
d_l(i) &= (\Delta w(l,i) - w_i)q_l(i) + \sum_k m_{kl}(x_i, x_-)(d_k(i-1) + q_k(i-1)w_{i-1}) && (32) \\
&= (\Delta w(l,i) - w_i + w_{i-1}) \cdot q_l(i) + \sum_k m_{kl}(x_i, x_-) \cdot d_k(i-1) && (33)
\end{aligned}
$$

Summing over $l$ gives the update rule for $w_i$:

$$
w_i = w_{i-1} + \sum_l \Delta w(l,i) \cdot q_l(i) + \sum_{k,l} m_{kl}(x_i, x_-) \cdot d_k(i-1) \qquad (34)
$$

The last, $d$-dependent sum is nonzero only if spikes arrive. It occurs because a new spike changes the probability estimates of previous states, and thereby the desired weight.

### 3.4  Summary of the learning algorithm

To simplify notation, we combined the pre- and postsynaptic Markov models into a single one. How does the learning rule look in terms of the original pre- and postsynaptic parameters? If the presynaptic model has $N^{pre}$ states and the postsynaptic one $N^{post}$, then the combined model has $N^{pre} \cdot N^{post}$ states. At each time step, we have to update not only the weight $w_i$ but $N^{pre} \cdot N^{post}$ signal traces $d$, which we will now write as $d_{gk}(i)$, where $g$ denotes the presynaptic and $k$ the postsynaptic state. However, one needs to update only $N^{pre} + N^{post}$ of the signal traces $q$, because they factorize into a pre- and a postsynaptic part: $q_{gk}(i) = q_g^{pre}(i) \cdot q_k^{post}(i)$. The learning algorithm is then given by:

- Initialization $(i = 0)$: Define the states and the parameter $e$ and $a$ of the pre- and postsynaptic Markov model.
  Define the weight change $\Delta w\left(\pi_i^{pre}, \pi_i^{post}, i\right)$ for all possible state pairs.
  Find the leading eigenvector $o$ of both Markov chains in the absence of spikes:

$$
\lambda^{pre} \cdot o_l^{pre} = \sum_k e_k^{pre}(0) \cdot a_{lk}^{pre} \cdot o_k^{pre} \qquad (35)
$$

  Initialize $w$, $d$, and $q$ ($w = w_0$; $d = 0$; $q = 1$ for arbitrary start state and 0 otherwise)

- Recursion ($i = 1, 2, ...$):

$$n^{pre} = \left( \sum_{kl} q_k^{pre} \cdot e_l^{pre}(x_i^{pre}) \cdot a_{kl}^{pre} \cdot o_l^{pre} / (\lambda^{pre} \cdot o_k^{pre}) \right)^{-1} \quad (36)$$

$$m_{kl}^{pre} = n^{pre} \cdot e_l^{pre}(x_i^{pre}) \cdot a_{kl}^{pre} \cdot o_l^{pre} / (\lambda^{pre} \cdot o_k^{pre}) \quad (37)$$

$$q_l^{pre} \leftarrow \sum_k m_{kl}^{pre} \cdot q_k^{pre} \quad (38)$$

and analogous equations for $n^{post}$, $m^{post}$, and $q^{post}$.

$$dw = \sum_{h,l} \Delta w(h,l,i) \cdot q_h^{pre} \cdot q_l^{post} + \sum_{h,l} \sum_{g,k} m_{gh}^{pre} \cdot m_{kl}^{post} \cdot d_{gk} \quad (39)$$

$$d_{hl} \leftarrow (\Delta w(h,l,i) - dw) \cdot q_h^{pre} \cdot q_l^{post} + \sum_{g,k} m_{gh}^{pre} \cdot m_{kl}^{post} \cdot d_{gk} \quad (40)$$

$$w \leftarrow w + dw \quad (41)$$

- Terminate at the end of the spike sequences $x^{pre}$ and $x^{post}$.

## 4 Conclusion

This demonstrates that the basic principle of associating not individual spikes, but whole firing episodes, can be implemented in a causal learning rule, which depends only on past signals. This rule does not have to store the time of all past spikes, but only a few signal traces $q$ and $d$, and may thus be biologically plausible. For the right parameter choice, it agrees well with some nonlinear features of cortical synaptic plasticity (fig. 2). This does not imply that actual synaptic plasticity follows the same rule, but only that these particular features are consistent with our basic principle. Based on the predictions of this rule, one could design more precise experimental tests of whether cortical synaptic plasticity associates individual spikes or whole firing episodes.

### Acknowledgments

This work was supported by R01-EY11001. We thank T. Sejnowski for his comments on a similar type of learning rules, which he suggested to call "hidden Hebbian learning". The second author (KM) would like to emphasize that his contribution to this paper was limited to assistance in writing.

## References

[1] G.-Q. Bi and M.-M. Poo Synaptic modification by correlated activity: Hebb's postulate revisited. *Ann. Rev. Neurosci.*, 24:139–166, 2001.

[2] O. Paulsen and T. J. Sejnowski. Natural patterns of activity and long-term synaptic plasticity. *Curr Opin Neurobiol.*, 10:172–179, 2000.

[3] W. Senn, H. Markram, and M. Tsodyks. An algorithm for modifying neurotransmitter release probability based on pre- and postsynaptic spike timing. *Neural Comput.*, 13:35–67, 2001.

[4] P. J. Sjostrom, Turrigiano G. G., and S. B. Nelson. Rate, timing, and cooperativity jointly determine cortical synaptic plasticity. *Neuron*, 32:1149–1164, 2001.

[5] R. C. Froemke and Y. Dan. Spike-timing-dependent synaptic modification induced by natural spike trains. *Nature*, 416:433–438, 2002.

[6] L. R. Rabiner. A tutorial on hidden Markov models and selected applications in speech recognition. *Proceedings of the IEEE*, 77:257–286, 1989.
